# Simplicial Mixtures of Markov Chains: Distributed Modelling of Dynamic User Profiles

**Mark Girolami**
Department of Computing Science
University of Glasgow
Glasgow, UK
girolami@dcs.gla.ac.uk

**Ata Kabán**
School of Computer Science
University of Birmingham
Birmingham, UK
a.kaban@cs.bham.ac.uk

## Abstract

To provide a compact generative representation of the sequential activity of a number of individuals within a group there is a tradeoff between the definition of individual specific and global models. This paper proposes a linear-time distributed model for finite state symbolic sequences representing traces of individual user activity by making the assumption that heterogeneous user behavior may be 'explained' by a relatively small number of common structurally simple behavioral patterns which may interleave randomly in a user-specific proportion. The results of an empirical study on three different sources of user traces indicates that this modelling approach provides an efficient representation scheme, reflected by improved prediction performance as well as providing low-complexity and intuitively interpretable representations.

## 1 Introduction

The now commonplace ability to accurately and inexpensively log the activity of individuals in a digital environment makes available a variety of traces of user activity and with it the necessity to develop efficient representations, or profiles, of individuals. Most often, such recordings take the form of streams of discrete symbols ordered in time. The modelling of time dependent sequences of discrete symbols employing $n$'th order Markov chains has been extensively studied in a number of domains. The representation provided by such models is global in the sense that it is assumed that one global generating process underlies all observed sequences. To capture the possible heterogeneous nature of the observed sequences a model with a number of differing generating processes needs to be considered. Indeed the notion of a heterogeneous population, characterized for example by occupational mobility and consumer brand preferences, has been captured in the *Mover-Stayer* model [3]. This model is a discrete time stochastic process that is a two component mixture of first order Markov chains, one of which is degenerate and possesses an identity transition matrix characterizing the *stayers* in the population. The original notion of a two-component mixture of Markov chains has recently been extended to the general form of a mixture model of Markov chains in [2]. Whilst the main motivation was the visualization of the class structure inherent in the browsing patterns of visitors to a commercial website, each class of users being characterized by their global behavior, such mixture models will

not be appropriate for identifying the shared behavioral patterns which are the basis of multiple relationships between users and groups of users and which may yield a more realistic model of the population.

The purpose of this paper is to develop a dynamic user model for individuals within a group that explicitly captures the assumption of the existence of a common set of behavioral patterns which can be estimated from all observed users along with their user-specific proportion of participation and these form the basis of individual profiles within a group. This is also a computationally attractive model, as simple structural characteristics may be assumed at the generative level, while allowing them to interleave randomly can account for more complex individual behavior. The resulting model is thus a distributed dynamic model which benefits from the recent technical developments in distributed parts based modelling of static vectorial data [7, 9, 5, 1, 8], with various applications including image decomposition, document modelling, information retrieval and collaborative filtering. Consistent generative semantics similar to the recently introduced latent Dirichlet allocation (LDA) [1] will be adopted and by analogy with [8] the resulting model will be referred to as a simplicial mixture.

## 2 Simplicial Mixtures of Markov Chains

Assume that a sequence of $L$ symbols $s_L s_{L-1}, \cdots, s_0$, denoted by $\mathbf{s}$, can be drawn from a dictionary $\mathcal{S}$ by a process $k$, which has initial state probability $P_1(k)$ and has $|\mathcal{S}|^{m+1}$ state transition probabilities denoted by $T(s_m, \cdots, s_1 \rightarrow s_0|k)$. The number of times that the symbol $s_0$ follows from the state defined by the $m$-tuple of symbols $s_m, \cdots, s_1$ within the $n$-th sequence is denoted as $r_n^{s_m,\cdots,s_1 \rightarrow s_0}$ and so the probability of the sequence of symbols under the $k$'th $m$-th order Markov process is $P(\mathbf{s}|k) = P_1(k) \prod_{s_m=1}^{|\mathcal{S}|} \cdots \prod_{s_0=1}^{|\mathcal{S}|} T(s_m, \cdots, s_1 \rightarrow s_0|k)^{r^{s_m,\cdots,s_1 \rightarrow s_0}}$. To introduce a more compact notation we represent the elements of the state transition matrix for the $k$'th Markov process by $T_{m\cdots 0,k}$ and the counts $r^{s_m,\cdots,s_1 \rightarrow s_0}$ within the $n$'th observed sequence as $r_n^{m\cdots 0}$. In addition, we employ `Start` and `Stop` states in each symbol sequence $\mathbf{s}_n$ and incorporate the initial state distribution of the `Start` state as the transition probabilities from this state within the state transition matrix $\mathrm{T}_k$. We denote the set of all state transition matrices $\{\mathrm{T}_1, \cdots, \mathrm{T}_k, \cdots, \mathrm{T}_K\}$ as $\mathbf{T}$. Suppose that we are given a set of symbolic trajectories $\{\mathbf{s}_n\}_{n=1:N}$ over a common finite state space, each having length $L_n$. As opposed and somewhat complementary to cluster models for trajectories which try to model inter-sequence heterogeneities, our intuition is that sequences over a common finite state space, provided they are sufficiently long and possibly non-stationary, could have several randomly interleaved generator processes, some of which might be common to several sequences. To account for this idea, we will adopt a similar modelling strategy to LDA.

The complete generative semantics of LDA allows us to describe the process of sequence generation where mixing components $\boldsymbol{\lambda} = [\lambda_1, \cdots, \lambda_k, \cdots, \lambda_K]$ are $K$-dimensional Dirichlet random variables and so are drawn from the $K - 1$ dimensional simplex defined by the Dirichlet distribution $\mathcal{D}(\boldsymbol{\lambda}|\boldsymbol{\alpha})$ with parameters $\boldsymbol{\alpha}$. These are then combined with the individual state-transition probabilities $\mathrm{T}_k$, which are model parameters to be estimated, and yield the symbol transition probabilities $T_{m\cdots 0} = \sum_{k=1}^{K} T_{m\cdots 0,k}\lambda_k$. The overall probability for a sequence $\mathbf{s}_n$ under such a mixture, which we shall now refer to as a simplicial mixture [8], denoted as $P(\mathbf{s}_n|\mathbf{T}, \boldsymbol{\alpha})$ is equal to

$$\int_\triangle P(\mathbf{s}_n|\mathbf{T}, \boldsymbol{\lambda})\mathcal{D}(\boldsymbol{\lambda}|\boldsymbol{\alpha})d\boldsymbol{\lambda} = \int_\triangle d\boldsymbol{\lambda}\mathcal{D}(\boldsymbol{\lambda}|\boldsymbol{\alpha}) \prod_{s_m=1}^{|\mathcal{S}|} \cdots \prod_{s_0=1}^{|\mathcal{S}|} \left\{ \sum_{k=1}^{K} T_{m\cdots 0,k}\lambda_k \right\}^{r_n^{m\cdots 0}} \quad (1)$$

Each sequence will have its own expectation under the Dirichlet mixing coefficients and

so the ability of such a representation to model intra-sequence heterogeneity emerges naturally.

The following subsections briefly present the details of the identification of this model, which also highlights the close relationship between two existing related models, specifically the probabilistic latent semantic analysis (PLSA) [5] and LDA [1] as being instances of the same theoretical model and differing only in the estimation procedure adopted [4].

## 2.1 Parameter Estimation and Inference

Exact inference within the LDA framework is not possible [1], however the likelihood can be lower-bounded by introducing a sequence specific parameterised variational posterior $Q_n(\boldsymbol{\lambda})$ whose parameters will depend on $n$

$$\log P(\mathbf{s}_n | \mathbf{T}, \boldsymbol{\alpha}) \geq \mathsf{E}_{Q_n(\boldsymbol{\lambda})} \left[ \log \left\{ P(\mathbf{s}_n | \mathbf{T}, \boldsymbol{\lambda}) \frac{\mathcal{D}(\boldsymbol{\lambda} | \boldsymbol{\alpha})}{Q_n(\boldsymbol{\lambda})} \right\} \right] \qquad (2)$$

Where $\mathsf{E}_{Q_n(\boldsymbol{\lambda})}$ denotes expectation with respect to $Q_n(\boldsymbol{\lambda})$. The bound can be defined using the *Maximum a Posteriori* (MAP) estimator, such that $Q_n(\boldsymbol{\lambda}) = \delta(\boldsymbol{\lambda} - \boldsymbol{\lambda}_n^{MAP})$, in which case (2) is equal to $\log P(\mathbf{s}_n | \mathbf{T}, \boldsymbol{\lambda}_n^{MAP}) + \log \mathcal{D}(\boldsymbol{\lambda}_n^{MAP} | \boldsymbol{\alpha}) + \mathcal{H}^\delta$ where $\mathcal{H}^\delta$ denotes the entropy of the delta function around $\boldsymbol{\lambda}_n^{MAP}$ (which can be discarded in this setting as it does not depend on the model parameters, although it amounts to minus infinity). Forming a Lagrangian from the above to enforce the constraint that $\boldsymbol{\lambda}^{MAP}$ is a sample point from a Dirichlet variable then taking derivatives with respect to the $\lambda_k^{MAP}$, a convergent series of updates $\lambda_{kn}^t$ is obtained where the superscript denotes the $t$'th iteration. As in [7], for each observed sequence in the sample a MAP value for the variable $\boldsymbol{\lambda}$ is iteratively estimated by the following multiplicative updates

$$\tilde{\lambda}_{kn} = (\alpha_k - 1) + \lambda_{kn}^t \sum_{s_m=1}^{|\mathcal{S}|} \cdots \sum_{s_0=1}^{|\mathcal{S}|} r_n^{m\cdots 0} \frac{T_{m\cdots 0, k}}{\sum_{l=1}^K T_{m\cdots 0, l} \lambda_{ln}^t} \quad ; \quad \lambda_{kn}^{t+1} = \frac{\tilde{\lambda}_{kn}}{L_n + \sum_k (\alpha_k - 1)} \qquad (3)$$

where $L_n = \sum_{s_m \cdots s_0} r_n^{m\cdots 0}$ is the length of the sequence $\mathbf{s}_n$. Once the MAP values $\boldsymbol{\lambda}_n^{MAP}$ for each $\mathbf{s}_n$ are obtained a similar multiplicative iteration for the transition probabilities can be obtained

$$\tilde{T}_{m\cdots 0, k} = T_{m\cdots 0, k}^t \sum_{n=1}^N r_n^{m\cdots 0} \frac{\lambda_{kn}^{MAP}}{\sum_{l=1}^K T_{m\cdots 0, l}^t \lambda_{ln}^{MAP}} \quad ; \quad T_{m\cdots 0, k}^{t+1} = \frac{\tilde{T}_{m\cdots 0, k}}{\sum_{s_0'=1}^{|\mathcal{S}|} \tilde{T}_{m\cdots 0', k}} \qquad (4)$$

The final parameter is that of the prior Dirichlet distribution, maximum likelihood estimation yields the estimated distribution parameters $\boldsymbol{\alpha}$ given the $\boldsymbol{\lambda}_n^{MAP}$ [6, 1]. Note that both (3) and (4) require an elementwise matrix multiplication and division so these iterations will scale linearly with the number of non-zero state-transition counts. It is interesting to note that the MAP estimator under a uniform Dirichlet distribution exactly recovers the *aspect mixture model* of [5] as a special case of the MAP estimated LDA model.

### 2.1.1 Variational Parameter Estimation and Inference

While being optimal in analyzing an existing data set, MAP estimators are notoriously prone to overfitting, especially where there is a paucity of available data [10] and so the variational Bayes (VB) approach detailed in [1] can be adopted by considering $Q_n(\boldsymbol{\lambda}) = \mathcal{D}(\boldsymbol{\lambda} | \boldsymbol{\gamma}_n)$, where $\boldsymbol{\gamma}_n$ is a sequence-specific variational free parameter vector. The above (2) can be further lower-bounded by noting that

$$\log P(\mathbf{s}_n | \mathbf{T}, \boldsymbol{\lambda}) \geq \sum_{s_m=1}^{|\mathcal{S}|} \cdots \sum_{s_0=1}^{|\mathcal{S}|} \sum_{k=1}^K r_n^{m\cdots 0} Q_{m\cdots 0, n}(k) \log \left\{ \lambda_k \frac{T_{m\cdots 0, k}}{Q_{m\cdots 0, n}(k)} \right\} \qquad (5)$$

where $\sum_k Q_{m\cdots 0,n}(k) = 1$, $Q_{m\cdots 0,n}(k) \geq 0$ are additional variational parameters. Alternatively, $Q_{m\cdots 0,n}(.)$ can also be understood as a variational distribution on a discrete hidden variable with $K$ possible outcomes that selects which transition matrix is active at each time step of the generative process.

Replacing (5) in (2), expanding and evaluating $\mathsf{E}_{\mathcal{D}(\boldsymbol{\lambda}|\boldsymbol{\gamma}_n)}[\log \lambda_k] = \psi(\gamma_k) - \psi(\sum_{k'} \gamma_{k'})$, where $\psi$ denotes the digamma function, then solving for $Q_{m\cdots 0,n}(k)$ and $\gamma_{kn}$ and finally combining yields the following multiplicative iterative update for the sequence specific variational free parameter $\boldsymbol{\gamma}_n$

$$\gamma_{kn}^{t+1} = \alpha_k + \exp\{\psi(\gamma_{kn}^t)\} \sum_{s_m=1}^{|\mathcal{S}|} \cdots \sum_{s_0=1}^{|\mathcal{S}|} r_n^{m\cdots 0} \frac{T_{m\cdots 0,k}}{\sum_{k'=1}^K T_{m\cdots 0,k'} \exp\{\psi(\gamma_{k'n}^t)\}} \quad (6)$$

Solving for the transition probabilities and combining with the fixed point solutions for each $Q_{m\cdots 0,n}(k)$ yields the following

$$\tilde{T}_{m\cdots 0,k} = T_{m\cdots 0,k}^t \sum_{n=1}^N r_n^{m\cdots 0} \frac{\exp\{\psi(\gamma_{kn}^t)\}}{\sum_{k'=1}^K T_{m\cdots 0,k'}^t \exp\{\psi(\gamma_{k'n}^t)\}} \; ; \quad T_{m\cdots 0,k}^{t+1} = \frac{\tilde{T}_{m\cdots 0,k}}{\sum_{s_0'} \tilde{T}_{m\cdots 0',k}}$$
(7)

As before the parameters of the prior Dirichlet distribution $\boldsymbol{\alpha}$ given the variational parameters $\boldsymbol{\gamma}_n$ are estimated using standard methods [6, 1].

## 2.2 Prediction with Simplicial Mixtures

The predictive probability of observing symbol $s_{next}$ given a sequence of $L$ symbols $\mathbf{s}_n = \{s_{L_n}, \cdots, s_1\}$ is given as $P(s_{next}|\mathbf{s}_n) = \mathsf{E}_{P(\boldsymbol{\lambda}|\mathbf{s}_n)}\{P(s_{next}|s_m \cdots s_1, \boldsymbol{\lambda})\} \approx \sum_{k=1}^K T(s_{next}|s_m \cdots s_1, k)\mathsf{E}_{Q_n(\boldsymbol{\lambda})}\{\lambda_k\}$. It should be noted that while $m$-th order Markov chains form the basis of the representation, the resulting simplicial mixture is not $m$-th order Markov with any global transition model. Rather it approximates the individual $m$-th order models while keeping the generative parameter set compact. The $m$-th order information of each individual's past behaviour is embodied in the individual-specific latent variable estimate. On the other hand in a mixture model one component is responsible for sequence generation so within a cluster the representation is still global $m$-th order. Employing the MAP approximation for the Dirichlet distribution then $\mathsf{E}_{Q_n(\boldsymbol{\lambda})}\{\lambda_k\} = \mathsf{E}_{\delta(\boldsymbol{\lambda}-\boldsymbol{\lambda}_n^{MAP})}\{\lambda_k\} = \lambda_{kn}^{MAP}$ where $\lambda_{kn}^{MAP}$ is the $k$-th dimension of $\boldsymbol{\lambda}_n^{MAP}$. Employing the variational Dirichlet approximation then $\mathsf{E}_{Q_n(\boldsymbol{\lambda})}\{\lambda_k\} = \mathsf{E}_{\mathcal{D}(\boldsymbol{\lambda}|\boldsymbol{\gamma}_n)}\{\lambda_k\} = \gamma_{kn}/\sum_{l=1}^K \gamma_{ln}$ therefore given a new sequence $\mathbf{s}_{new}$, the symbol $s_{next}$ which is most likely to be predicted from the model as a suggested continuation of the sequence, is the maximum argument of $P(s_{next}|\mathbf{s}_n)$.

# 3 Distributed Modelling of Dynamic Profiles

## 3.1 Datasets

### 3.1.1 Telephone Usage Modelling

The ability to model the usage of a telephone service is of importance at a number of levels, e.g. to obtain a predictive model of customer specific activity and service usage for the purposes of service provision planning, resource management of switching capacity, identification of fraudulent usage of services. A representative description can be based on the distribution of the destination numbers dialled and connected by the customer, in which

case a multinomial distribution over the dialling codes can be employed. One method of encoding the destination numbers dialled by a customer is to capture the geographic location of the destination, or the mobile service provider if not a land based call. This is useful in determining the potential demand placed on telecommunication switches which route traffic from various geographical regions on the service providers network. Two weeks of transactions from a UK telecommunications operator were logged during weekdays, amounting to 36,492,082 and 45,350,654 transactions in each week respectively. All transactions made by commercial customers in the Glasgow region of the UK were considered in this study. This amounts to 1,172,578 transactions from 12,202 high usage customers in the first week considered and 1,753,304 transactions being made in the following week. The mapping from dialling number to geographic region or mobile operator was encoded with 87 symbols amounting to a possible 7,569 symbol transitions. Each customers activity is defined by a sequence of symbols defining the sequence of calls made over each period considered and these are employed to encode activity in a customer specific generative representation.

### 3.1.2 Web Page Browsing

The second data set used in this study is a selected subset of the msnbc.com user navigation collection employed in [2]. Sequences of users who visited at least 9 of the overall 17 page categories (frontpage, news, tech, local, opinion, on-air, misc,weather, msn-news, health, living, business, msn-sports, sports, summary, bbs, travel) have been retained, this selection criteria is motivated by the observation that there would be little scope in trying to model interleaved dynamic behavior in observables which are too short to reveal any intra-sequence heterogeneity. The resulting data set, referred to as WEB, totals 119,667 page requests corresponding to 1,480 web browsing sessions.

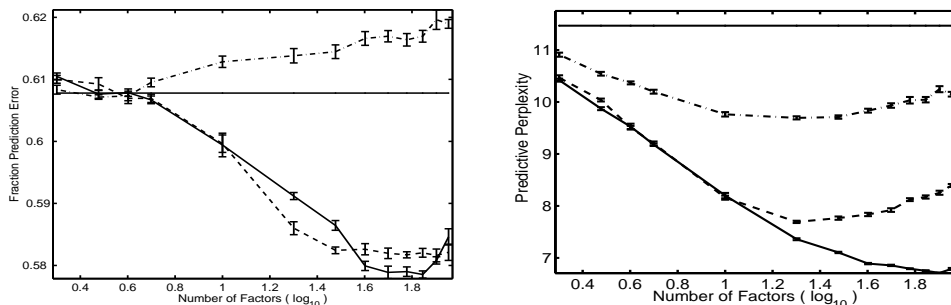

Figure 1: Left: percentage of incorrect predictions against the number of model factors; right: predictive perplexity of each model against model order for the PHONE dataset. Solid straight line: global first order MC, dash: MAP estimated simplicial mixture, solid line: VB estimated simplicial mixture, dash-dot: mixture model.

### 3.2 Results

In each experiment the objective assessment of model performance is evaluated by the predictive perplexity, $\exp\{-1/N \sum_{m=1}^{N_{test}} \log P(s_{next}|s_m)\}$. In addition, the predictive accuracy of all models is measured under a 0-1 loss. Given a number of previously unobserved truncated sequences, the number of times the model correctly predicts the symbol which follows in the sequence is then counted. In all mixture models naive random initialization of the parameters was employed and parameter estimation was halted when the

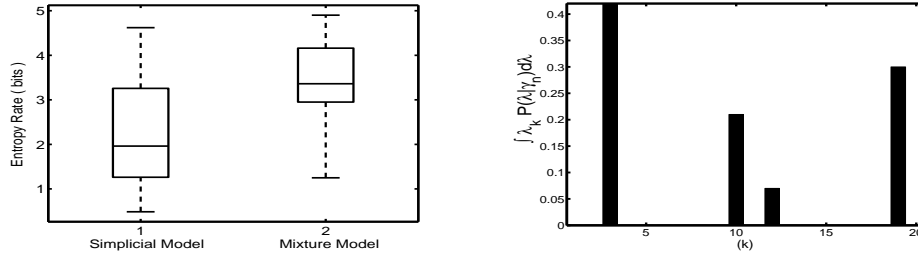

Figure 2: Left: distribution of entropy rates for the transition matrices of a 20-factor mixture and simplicial mixture models (VB). Right: the expected value of the Dirichlet variable under the variational approximation for one customer indicating the levels of participation in factor specific behaviors.

in-sample likelihood did not improve by more than 0.001%, no annealing or early stopping was utilized, fifteen randomly initialized parameter estimation runs for each model were performed. The number of mixture components for the models ranged from 2 up to 200. On the PHONE data set the parameters of a global first-order Markov chain (bigram), mixtures of Markov chains [2], and simplicial mixtures of Markov chains (using both the MAP and VB estimation procedures) are estimated using the first week of customer transactions and the predictive capabilities of the models are assessed on the transactions from the following week. The results are summarized in Figure 1, from the predictive perplexity measures it is clear that the simplicial representation provides a statistically (tested at the 5% level using a Wilcoxon Rank Sum test) and practically significant reduction in perplexity over the global and mixture models. This is also reflected in the levels of prediction error under each model, however the mixture models tend to perform slightly worse than the global model. As expected the MAP estimated simplicial model performs slightly worse than that obtained using VB [1]. This also provides an additional insight as to why LDA models improve upon PLSA, as they are in fact both the same model using different approximations to the likelihood, refer to [10] for an illustrative discussion on the weaknesses of MAP estimators. As a comparison to different structural models hidden Markov models with a range of hidden states were also tested on this data set the best results obtained were for a ten state model which achieved a predictive perplexity score of (mean±standard-deviation) $11.119 \pm 0.624$ and fraction prediction error of $0.674 \pm 0.959$, considerably poorer than that obtained by the models considered here.

In addition to the predictive capability of a simplicial representation of a customers activity the cost of encoding such a representation can be assessed by measuring the entropy rate of each of the constituent transition matrices which act as a basis in the representation of the individual specific generative process. The left hand plot of Figure (2) shows the distribution of the entropy rates for the transition probabilities in twenty factor simplicial and mixture models, the results are obtained from fifty randomly initialized estimation procedures. The entropy rates for the simplicial mixture are significantly lower than that of a mixture model indicating that the basis of each representation describes a number of simpler processes.

The final experiment demonstrated considers the WEB data set. The results of ten-fold cross-validated predictive perplexities again show statistically significant improvement obtained with the VB-estimated simplicial mixture (again tested using the ranksum Wilcoxon test at the 5% level). The results are summarized in Figure 3. Five of the estimated transition factors of a twenty-factor model are shown in Figure 4, demonstrating once more that the proposed model creates a low entropy and an easily interpretable dynamic factorial representation. The numbers on the axes on these charts correspond to the 17 page cat-

egories enumerated earlier and the average strength of each of these factors amongst the full set of twenty factors computed as $\frac{1}{N} \sum_{n=1}^{N} \mathsf{E}_{\mathcal{D}(\lambda|\gamma_n)}\{\lambda_k\}$ is also given above each chart. We can see that a behavioral feature manifested is a keen interest to visit pages about 'news' along with a quite dynamic transition model (left hand chart) which characterizes around 12% of the behavioral patterns of the entire user population under consideration while static state-repetition (second chart) or an almost exclusive interest in viewing the homepage (last chart) etc represent also relatively strong common characteristics of browsing behavior. The distribution of the entropy rates of the full set of these twenty basis-transitions in comparison to those obtained from the mixture model is given on the right hand plot of Figure 3. Clearly, the coding efficiency of a simplicial mixture representation is significantly (statistically tested) superior. Note also these basis-transitions embody correlated transitions (transitions which appear in similar dynamical contexts and so have similar functionality), as can be seen from the multiplicative nature of the equations used for identifying the model. It is not surprising then that state repetitions or transitions which express focused interest in one of the topic categories appear together on distinct factors. We can also see a joint interest in msnnews and msnsport being present together on the 4-th chart of Figure 4 — indeed, as the prefix of these page categories also indicates, these are related page categories.

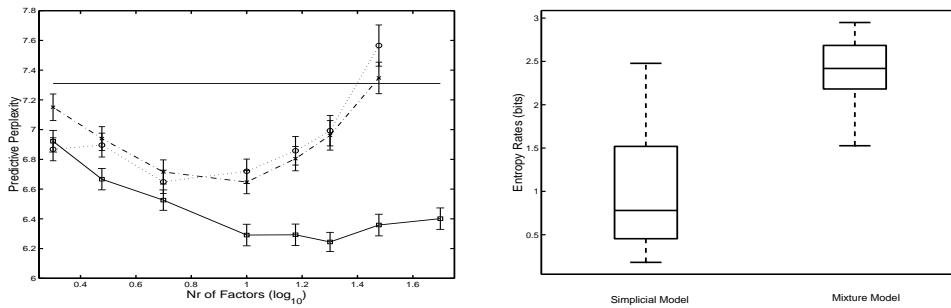

Figure 3: Left: the predictive perplexity for the WEB data (straight line: global first-order Markov chain, dash-dot: mixture of Markov chains, dotted line: simplicial mixture estimated by MAP, solid line: simplicial mixture estimated by VB). Right: the distribution of entropy rates.

## 4  Conclusions

This paper has presented a linear time method to model finite-state sequences of discrete symbols which may arise from user or customer activity traces. The main feature of the proposed approach has been the assumption that heterogeneous user behavior may be 'explained' by the interleaved action of some structurally simple common generator processes.

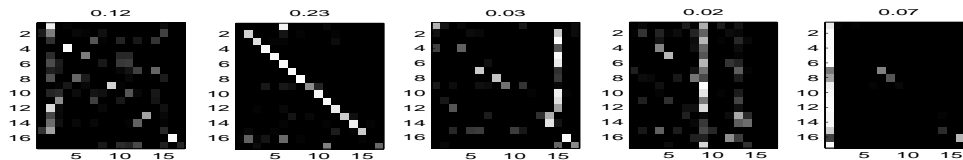

Figure 4: State transition matrices of selected factors from a 20-factor run on WEB.

An empirical study has been conducted on two *real-world* collections of user activity which has demonstrated this to be an efficient representation, revealed by both objective measures of prediction performance, low entropy rates, and interpretable representations of the user profiles provided.

## Acknowledgements

Mark Girolami is part of the DETECTOR project funded by the Department of Trade and Industry (DTI) Management of Information (LINK) Programme and the Engineering & Physical Sciences Research Council (EPSRC) grant GR/R55184.

## References

[1] D. M. Blei, A. Y. Ng & M. I. Jordan, *Latent Dirchlet Allocation*, Journal of Machine Learning Research, 3(5):993–1022, 2003.

[2] I. Cadez, D. Heckerman, C. Meek, P. Smyth & S. White, *Model-based clustering and visualisation of navigation patterns on a web site*, Journal of data Mining and Knowledge Discovery, in press.

[3] H. Frydman, *Maximum likelihood estimation in the mover-stayer model*, Journal of the American Statistical Society, 79, 632-638, 1984.

[4] M. Girolami and A. Kabán, *On an equivalence between PLSI and LDA*, Proc. 26-th Annual International ACM SIGIR Conference, 2003, pp. 433–434.

[5] T. Hofmann,*Unsupervised learning by probabilistic latent semantic analysis*, Machine Learning, 42, 177-196, 2001.

[6] G. Ronning, *Maximum likelihood estimation of Dirichlet distributions*, Journal of Statistical Computation and Simulation, 32:4, 215-221, 1989.

[7] D. Lee & H. Sebastian Seung, *Algorithms for Non-negative Matrix Factorization*, Advances in Neural Information Processing Systems 13, ed's Leen, Todd K, Dietterich, Thomas G. and Tresp, Volker, 556–562, MIT Press, 2001.

[8] T. Minka & J. Lafferty, *Expectation-propogation for the generative aspect model*, Proceedings of the Eighteenth Conference on Uncertainty in Artificial Intelligence, 2002.

[9] D. A. Ross & R. S. Zemel, *Multiple-cause vector quantiztion*, Advances in Neural Information Processing Systems 15, 2003.

[10] H. Lappalainen & J. W. Miskin. *Ensemble Learning*. In M. Girolami, editor, Advances in Independent Component Analysis, 75-92, Springer-Verlag, 2000.
